# Geometric Clustering using the Information Bottleneck method

**Susanne Still**
Department of Physics
Princeton Unversity, Princeton, NJ 08544
`susanna@princeton.edu`

**William Bialek**
Department of Physics
Princeton Unversity, Princeton, NJ 08544
`wbialek@princeton.edu`

**Léon Bottou**
NEC Laboratories America
4 Independence Way, Princeton, NJ 08540
`leon@bottou.org`

## Abstract

We argue that K–means and deterministic annealing algorithms for geometric clustering can be derived from the more general Information Bottleneck approach. If we cluster the identities of data points to preserve information about their location, the set of optimal solutions is massively degenerate. But if we treat the equations that define the optimal solution as an iterative algorithm, then a set of "smooth" initial conditions selects solutions with the desired geometrical properties. In addition to conceptual unification, we argue that this approach can be more efficient and robust than classic algorithms.

## 1   Introduction

Clustering is one of the most widespread methods of data analysis and embodies strong intuitions about the world: Many different acoustic waveforms stand for the same word, many different images correspond to the same object, etc.. At a colloquial level, clustering groups data points so that points within a cluster are more similar to one another than to points in different clusters. To achieve this, one has to assign data points to clusters and determine how many clusters to use. (Dis)similarity among data points might, in the simplest example, be measured with the Euclidean norm, and then we could ask for a clustering of the points[1] $\{\mathbf{x}_i\}$, $i = 1, 2, ..., N$, such that the mean square distance among points within the clusters is minimized,

$$\frac{1}{N_c} \sum_{c=1}^{N_c} \frac{1}{n_c} \sum_{ij \in c} |\mathbf{x}_i - \mathbf{x}_j|^2, \tag{1}$$

where there are $N_c$ clusters and $n_c$ points are assigned to cluster $c$. Widely used iterative reallocation algorithms such as K–means [5, 8] provide an approximate solution to the

problem of minimizing this quantity. Several alternative cost functions have been proposed (see e.g. [5]), and some use analogies with physical systems [3, 7]. However, this approach does not give a principled answer to how many clusters should be used. One often introduces and optimizes another criterion to find the optimal number of clusters, leading to a variety of "stopping rules" for the clustering process [5]. Alternatively, cross-validation methods can be used [11] or, if the underlying distribution is assumed to have a certain shape (mixture models), then the number of clusters can be found, e.g by using the BIC [4].

A different view of clustering is provided by information theory. Clustering is viewed as lossy data compression; the identity of individual points ($\sim \log_2 N$ bits) is replaced by the identity of the cluster to which they are assigned ($\sim \log_2 N_c$ bits $\ll \log_2 N$ bits). Each cluster is associated with a representative point $\mathbf{x}_c$, and what we lose in the compression are the deviations of the individual $\mathbf{x}_{i \in c}$, from the representative $\mathbf{x}_c$. One way to formalize this trading between data compression and error is rate–distortion theory [10], which again requires us to specify a function $d(\mathbf{x}_i, \mathbf{x}_c)$ that measures the magnitude of our error in replacing $\mathbf{x}_i$ by $\mathbf{x}_c$. The trade-off between the coding cost and the distortion defines a one parameter family of optimization problems, and this parameter can be identified with temperature through an analogy with statistical mechanics [9]. As we lower the temperature there are phase transitions to solutions with more and more distinct clusters, and if we fix the number of clusters and vary the temperature we find a smooth variation from "soft" (probabilistic) to "hard" (deterministic) clustering. For distortion functions $d(\mathbf{x}, \mathbf{x}') \propto (\mathbf{x} - \mathbf{x}')^2$, a deterministic annealing approach to solving the variational problem converges to the K–means algorithm in the limit of zero temperature [9].

A more general information theoretic approach to clustering, the Information Bottleneck method [13], explicitly implements the idea that our analysis of the data typically is motivated by our interest in some derived quantity (e.g., words from sounds) and that we should preserve this *relevant information* rather than trying to guess at what metric in the space of our data will achieve the proper feature selection. We imagine that each point $\mathbf{x}_i$ occurs together with a corresponding variable $\mathbf{v}_i$, and that $\mathbf{v}$ is really the object of interest.[2] Rather than trying to select the important features of similarity among different points $\mathbf{x}_i$, we cluster in $\mathbf{x}$ space to compress our description of these points while preserving as much information as possible about $\mathbf{v}$, and again this defines a one parameter family of optimization problems. In this formulation there is no need to define a similarity (or distortion) measure; this measure arises from the optimization principle itself. Furthermore, this framework allows us to find the optimal number of clusters for a finite data set using perturbation theory [12]. The Information Bottleneck principle thus allows a full solution of the clustering problem.

The Information Bottleneck approach is attractive precisely because the generality of information theory frees us from a need to specify in advance what it means for data points to be similar: Two points can be clustered together if this merger does not lose too much information about the relevant variable $\mathbf{v}$. More precisely, because mutual information is invariant to any invertible transformation of the variables, approaches which are built entirely from such information theoretic quantities are independent of any arbitrary assumptions about what it means for two points to be close in the data space. This is especially attractive if we want the same information theoretic principles to apply both to the analysis of, for example, raw acoustic waveforms and to the sequences of words for which these sounds might stand [2]. On the other hand, it is not clear how to incorporate a geometric intuition into the Information Bottleneck approach.

A natural and purely information theoretic formulation of geometric clustering might ask that we cluster the points, compressing the data index $i \in [1, N]$ into a smaller set of cluster

indices $c \in [1, N_c]$ so that we preserve as much information as possible about the *locations* of the points, i.e. location $\mathbf{x}$ becomes the relevant variable. Because mutual information is a geometric invariant, however, such a problem has an infinitely degenerate set of solutions. We emphasize that this degeneracy is a matter of principle, and not a failing of any approximate algorithm for solving the optimization problem. What we propose here is to lift this degeneracy by choosing the initial conditions for an iterative algorithm which solves the Information Bottleneck equations. In effect our choice of initial conditions expresses a notion of smoothness or geometry in the space of the $\{\mathbf{x}_i\}$, and once this is done the dynamics of the iterative algorithm lead to a finite set of fixed points. For a broad range of temperatures in the Information Bottleneck problem the solutions we find in this way are precisely those which would be found by a K–means algorithm, while at a critical temperature we recover the deterministic annealing approach to rate–distortion theory. In addition to the conceptual attraction of connecting these very different approaches to clustering in a single information theoretic framework, we argue that our approach may have some advantages of robustness.

## 2 Derivation of K–means from the Information Bottleneck method

We use the Information Bottleneck method to solve the geometric clustering problem and compress the data indices $i$ into cluster indices $c$ in a lossy way, keeping as much information about the location $\mathbf{x}$ in the compression as possible. The variational principle is then

$$\max_{p(c|i)} \left[ I(\mathbf{x}, c) - \lambda I(c, i) \right] \tag{2}$$

where $\lambda$ is a Lagrange parameter which regulates the trade-off between compression and preservation of relevant information. Following [13], we assume that $p(\mathbf{x}|i, c) = p(\mathbf{x}|i)$, i.e. the distribution of locations for a datum, if the index of the datum is known, does not depend explicitly on how we cluster. Then $p(\mathbf{x}|c)$ is given by the Markov condition

$$p(\mathbf{x}|c) = \frac{1}{p(c)} \sum_i p(\mathbf{x}|i) p(c|i) p(i). \tag{3}$$

For simplicity, let us discretize the space that the data live in, let us assume that it is a finite domain and that we can estimate the probability distribution $p(\mathbf{x})$ by a normalized histogram. Then the data we observe determine

$$p(\mathbf{x}|i) = \delta_{\mathbf{x}\mathbf{x}_i}, \tag{4}$$

where $\delta_{\mathbf{x}\mathbf{x}_i}$ is the Kronecker-delta which is 1 if $\mathbf{x} = \mathbf{x}_i$ and zero otherwise. The probability of indices is, of course, $p(i) = 1/N$.

The optimal assignment rule follows from the variational principle (2) and is given by

$$p(c|i) = \frac{p(c)}{Z(i, \lambda)} \exp \left[ \frac{1}{\lambda} \sum_{\mathbf{x}} p(\mathbf{x}|i) \log_2 \left[ p(\mathbf{x}|c) \right] \right]. \tag{5}$$

where $Z(i, \lambda)$ ensures normalization. This equation has to be solved self consistently together with eq.(3) and $p(c) = \sum_i p(c|i)/N$. These are the Information Bottleneck equations and they can be solved iteratively [13]. Denoting by $p_n$ the probability distribution

after the $n$-th iteration, the iterative algorithm is given by

$$p_n(c|i) = \frac{p_{n-1}(c)}{Z_n(i,\lambda)} \exp\left[\frac{1}{\lambda}\sum_{\mathbf{x}} p(\mathbf{x}|i)\log_2\left[p_{n-1}(\mathbf{x}|c)\right]\right], \tag{6}$$

$$p_n(\mathbf{x}|c) = \frac{1}{Np_{n-1}(c)}\sum_i p(\mathbf{x}|i)p_n(c|i), \tag{7}$$

$$p_n(c) = \frac{1}{N}\sum_i p_n(c|i). \tag{8}$$

Let $d(\mathbf{x},\mathbf{x}')$ be a distance measure on the data space. We choose $N_c$ cluster centers $\mathbf{x}_c^{(0)}$ at random and initialize

$$p_0(\mathbf{x}|c) = \frac{1}{Z_0(c,\lambda)}\exp\left[-\frac{1}{s}d(\mathbf{x},\mathbf{x}_c^{(0)})\right] \tag{9}$$

where $Z_0(c,\lambda)$ is a normalization constant and $s > 0$ is some arbitrary length scale – the reason for introducing $s$ will become apparent in the following treatment. After each iteration, we determine the cluster centers $\mathbf{x}_c^{(n)}$, $n \geq 1$, according to (compare [9])

$$0 = \sum_{\mathbf{x}} p_n(\mathbf{x}|c)\,\frac{\partial d(\mathbf{x},\mathbf{x}_c^{(n)})}{\partial \mathbf{x}_c^{(n)}}, \tag{10}$$

which for the squared distance reduces to

$$\mathbf{x}_c^{(n)} = \sum_{\mathbf{x}} \mathbf{x}\, p_n(\mathbf{x}|c). \tag{11}$$

We furthermore initialize $p_0(c) = 1/N_c$, where $N_c$ is the number of clusters. Now define the index $c_i^*$ such that it denotes the cluster with cluster center closest to the datum $\mathbf{x}_i$ (in the $n$-th iteration):

$$c_i^* := \arg\min_c d(\mathbf{x}_i,\mathbf{x}_c^{(n)}). \tag{12}$$

**Proposition:** If $0 < \lambda < 1$, and if the cluster indexed by $c_i^*$ is non–empty, then for $n \to \infty$

$$p(c|i) = \delta_{cc_i^*}. \tag{13}$$

**Proof:** From (7) and (4) we know that $p_n(\mathbf{x}|c) \propto \sum_i \delta_{\mathbf{x}\mathbf{x}_i} p_n(c|i)/p_{n-1}(c)$ and from (6) we have

$$p_n(c|i)/p_{n-1}(c) \propto \exp\left[\frac{1}{\lambda}\sum_{\mathbf{x}} p(\mathbf{x}|i)\log_2\left[p_{n-1}(\mathbf{x}|c)\right]\right], \tag{14}$$

and hence $p_n(\mathbf{x}|c) \propto (p_{n-1}(\mathbf{x}|c))^{1/\lambda}$. Substituting (9), we have $p_1(\mathbf{x}|c) \propto \exp\left[-\frac{1}{s\lambda}d(\mathbf{x},\mathbf{x}_c^{(0)})\right]$. The cluster centers $\mathbf{x}_c^{(n)}$ are updated in each iteration and therefore we have after $n$ iterations:

$$p_n(\mathbf{x}|c) \propto \exp\left[-\frac{1}{s\lambda^n}d(\mathbf{x},\mathbf{x}_c^{(n-1)})\right] \tag{15}$$

where the proportionality constant has to ensure normalization of the probability measure. Use (14) and (15) to find that

$$p_n(c|i) \propto p_{n-1}(c)\exp\left[-\frac{1}{s\lambda^n}d(\mathbf{x}_i,\mathbf{x}_c^{(n-1)})\right]. \tag{16}$$

and again the proportionality constant has to ensure normalization. We can now write the probability that a data point is assigned to the cluster nearest to it:

$$p_n(c_i^*|i) =$$

$$\left(1 + \frac{1}{p_{n-1}(c_i^*)} \sum_{c \neq c_i^*} p_{n-1}(c) \exp\left[-\frac{1}{s\lambda^n}\left(d(\mathbf{x}_i, \mathbf{x}_c^{(n-1)}) - d(\mathbf{x}_i, \mathbf{x}_{c_i^*}^{(n-1)})\right)\right]\right)^{-1} \quad (17)$$

By definition $d(\mathbf{x}_i, \mathbf{x}_c^{(n-1)}) - d(\mathbf{x}_i, \mathbf{x}_{c_i^*}^{(n-1)}) > 0 \;\forall c \neq c_i^*$, and thus for $n \to \infty$, $\exp\left[-\frac{1}{s\lambda^n}\left(d(\mathbf{x}_i, \mathbf{x}_c^{(n-1)}) - d(\mathbf{x}_i, \mathbf{x}_{c_i^*}^{(n-1)})\right)\right] \to 0$, and for clusters that do not have zero occupancy, i.e for which $p_{n-1}(c_i^*) > 0$, we have $p(c_i^*|i) \to 1$. Finally, because of normalization, $p(c \neq c_i^*|i)$ must be zero. $\square$

From eq. (13) follows with equations (4), (7) and (11) that for $n \to \infty$

$$\mathbf{x}_c = \frac{1}{n_c} \sum_{\mathbf{x}} \mathbf{x}_i \delta_{cc_i^*}, \quad (18)$$

where $n_c = \sum_i \delta_{cc_i^*}$. This means that for the square distance measure, this algorithm produces the familiar K–means solution: we get a hard clustering assignment (13) where each datum $i$ is assigned to the cluster $c_i^*$ with the nearest center. Cluster centers are updated according to eq. (18) as the average of all the points that have been assigned to that cluster. For some problems, the squared distance might be inappropriate, and the update rule for computing the cluster centers depends on the particular distance function (see eq. 10).

**Example.** We consider the squared Euclidean distance, $d(\mathbf{x}, \mathbf{x}') = |\mathbf{x} - \mathbf{x}'|^2/2$. With this distance measure, eq. (15) tells us that the (Gaussian) distribution $p(\mathbf{x}|c)$ contracts around the cluster center $\mathbf{x}_c$ as the number of iterations increases. The $\mathbf{x}_c$'s are, of course, recomputed in every iteration, following eq. (11).

We create a synthetic data set by drawing 2500 data points i.i.d. from four two-dimensional Gaussian distributions with different means and the same variance. Figure (1) shows the result of numerical iteration of the equations (14) and (16) – ensuring proper normalization – as well as (8) and (11), with $\lambda = 0.5$ and $s = 0.5$. The algorithm converges to a stable solution after $n = 14$ iterations.

This algorithm is less sensitive to initial conditions than the regular K–means algorithm. We measure the goodness of the classification by evaluating how much relevant information $I(\mathbf{x}, c)$ the solution captures. In the case we are looking at, the relevant information reduces to the entropy $H[p(c)]$ of the distribution $p(c)$ at the solution[3]. We used 1000 different random initial conditions for the cluster centers and for each, we iterated eqs. (8), (11), (14) and (16) on the data in Fig. 1. We found two different values for $H[p(c)]$ at the solution, indicating that there are at least two local maxima in $I(\mathbf{x}, c)$. Figure 2 shows the fraction of the initial conditions that converged to the global maximum. This number depends on the parameters $s$ and $\lambda$. For $d(\mathbf{x}, \mathbf{x}') = |\mathbf{x} - \mathbf{x}'|^2/2s$, the initial distribution $p^{(0)}(\mathbf{x}|c)$ is Gaussian with variance $s$. Larger variance $s$ makes the algorithm less sensitive to the initial location of the cluster centers. Figure 2 shows that, for large values of $s$, we obtain a solution that corresponds to the global maximum of $I(\mathbf{x}, c)$ for 100% of the initial conditions. Here, we fixed $\lambda$ at reasonably small values to ensure fast convergence ($\lambda \in \{0.05, 0.1, 0.2\}$). For these $\lambda$ values, the number of iterations till convergence lies

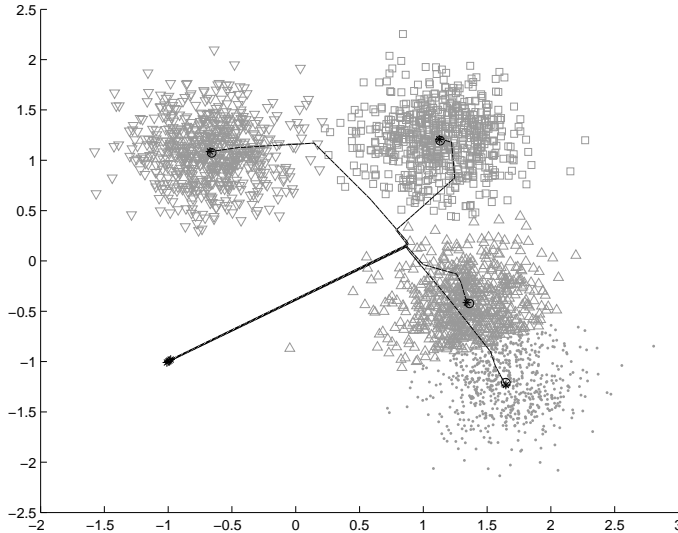

Figure 1: 2500 data points drawn i.i.d from four Gaussian distributions with different means and the same variance. Those data which got assigned to the same cluster are plotted with the same symbol. The dotted traces indicate movements of the cluster centers (black stars) from their initial positions in the lower left corner of the graph to their final positions close to the means of the Gaussian distributions (black circles) after 14 iterations.

between 10 and 20 (for $0.5 < s < 500$). As we increase $\lambda$ there is a (noisy) trend to more iterations. In comparison, we did the same test using regular K–means [8] and obtained a globally optimal solution from only 75.8% of the initial cluster locations.

To see how this algorithm performs on data in a higher dimensional space, we draw 2500 points from 4 twenty-dimensional Gaussians with variance 0.3 along each dimension. The typical euclidean distances between the means are around 7. We tested the robustness to initial center locations in the same way as we did for the two dimensional data. Despite the high signal to noise ratio, the regular K–means algorithm [8], run on this data, finds a globally optimal solution for only 37.8% of the initial center locations, presumably because the data is relatively scarce and therefore the objective function is relatively rough. We found that our algorithm converged to the global optimum for between 78.0% and 81.0% of the initial center locations for large enough values of $s$ ($1000 < s < 10000$) and $\lambda = 0.1$.

## 3 Discussion

**Connection to deterministic annealing.** For $\lambda = 1$, we obtain the solution

$$p_n(c|i) \propto \exp\left[-\frac{1}{s}d(\mathbf{x}_i, \mathbf{x}_c^{(n-1)})\right] \tag{19}$$

where the proportionality constant ensures normalization. This equation, together with eq. (11), recovers the equations derived from rate distortion theory in [9] (for square distance), only here the length scale $s$ appears in the position of the annealing temperature $T$ in [9]. We call this parameter the *annealing* temperature, because [9] suggests the following deterministic annealing scheme: start with large $T$; fix the $\mathbf{x}_c$'s and compute the optimal assignment rule according to eq. (19), then fix the assignment rule and compute the $\mathbf{x}_c$'s according to eq. (11), and repeat these two steps until convergence. Then lower the temper-

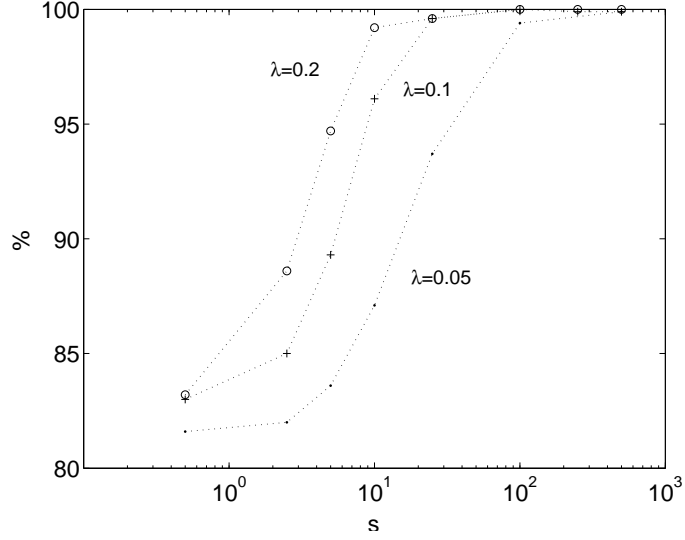

Figure 2: Robustness of algorithm to initial center positions as a function of the initial variance, $s$. 1000 different random initial positions were used to obtain clustering solutions on the data shown in Fig. 1. Displayed is, as a function of the initial variance $s$, the percent of initial center positions that converge to a global maximum of the objective function. In comparison, regular K–means [8] converges to the global optimum for only 75.8% of the initial center positions. The parameter $\lambda$ is kept fixed at reasonably small values (indicated in the plot) to ensure fast convergence (between 10 and 20 iterations).

ature and repeat the procedure. There is no general rule that tells us how slow the annealing has to be. In contrast, the algorithm we have derived here for $\lambda < 1$ suggests to start with a very large initial temperature, given by $s\lambda$, by making $s$ very large and to lower the temperature rapidly by making $\lambda$ reasonably small. In contrast to the deterministic annealing scheme, we do not iterate the equations for the optimal assignment rule and cluster centers till convergence before we lower the temperature, but instead the temperature is lowered by a factor of $\lambda$ after each iteration. This produces an algorithm that converges rapidly while finding a globally optimal solution with high probability.

For $\lambda = 1$, we furthermore find from eq. (15), that $p_n(\mathbf{x}|c) \propto \exp\left[-\frac{1}{s}d(\mathbf{x}, \mathbf{x}_c^{(n-1)})\right]$, and for $d(\mathbf{x}, \mathbf{x}') = |\mathbf{x} - \mathbf{x}'|^2/2$, the clusters are simply Gaussians.

For $\lambda > 1$, we obtain a useless solution for $n \to \infty$, that assigns all the data to one cluster.

**Optimal number of clusters**   One of the advancements that the approach we have laid out here should bring is that it should now be possible to extend our earlier results on finding the optimal number of clusters [12] to the problem of geometric clustering. We have to leave the details for a future paper, but essentially we would argue that as we observe a finite number of data points, we make an error in estimating the distribution that underlies the generation of these data points. This mis-estimate leads to a systematic error in evaluating the relevant information. We have computed this error using perturbation theory [12]. For deterministic assignments (as we have in the hard K–means solution), we know that a correction of the error introduces a penalty in the objective function for using more clusters and this allows us to find the optimal number of clusters. Since our result says that the penalty depends on the number of bins that we use to estimate the distribution underlying the data [12], we either have to know the resolution with which to look at our

data, or estimate this resolution from the size of the data set, as in e.g. [1, 6]. A combination of these insights should tell us how to determine, for geometrical clustering, the number of clusters that is optimal for a finite data set.

## 4   Conclusion

We have shown that it is possible to cast geometrical clustering into the general, information theoretic framework provided by the Information Bottleneck method. More precisely, we cluster the data keeping information about location and we have shown that the degeneracy of optimal solutions, which arises from the fact that the mutual information is invariant to any invertible transformation of the variables, can be lifted by the correct choice of the initial conditions for the iterative algorithm which solves the Information Bottleneck equations. We have shown that for a large range of values of the Lagrange multiplier $\lambda$ (which regulates the trade-off between compression and preservation of relevant information), we obtain an algorithm that converges to a hard clustering K–means solution. We have found some indication that this algorithm might be more robust to initial center locations than regular K–means. Our results also suggest an annealing scheme, which might prove to be faster than the deterministic annealing approach to geometrical clustering, known from rate–distortion theory [9]. We recover the later for $\lambda = 1$. Our results shed new light on the connection between the relatively novel Information Bottleneck method and earlier approaches to clustering, particularly the well-established K–means algorithm.

### Acknowledgments

We thank G. Atwal and N. Slonim for interesting discussions. S. Still acknowledges support from the German Research Foundation (DFG), grant no. Sti197.

## Footnotes

[1]Notation: All bold faced variables in this paper denote vectors.

[2]$\mathbf{v}$ does not have to live in the same space as the data $\mathbf{x}_i$.

[3] $I(\mathbf{x}, c) = H[p(c)] + \sum_{\mathbf{x}} p(\mathbf{x}) \sum_c p(c|\mathbf{x}) \log_2(p(c|\mathbf{x}))$. Deterministic assignments: $p(c|i) = \delta_{cc_i^*}$. Data points which are located at one particular position: $p(\mathbf{x}|i) = \delta_{\mathbf{x}\mathbf{x}_i}$. We thus have $p(c|\mathbf{x}) = \frac{1}{Np(c)} \sum_i p(c|i)p(\mathbf{x}|i) = \frac{1}{Np(c)} \sum_i \delta_{\mathbf{x}\mathbf{x}_i} \delta_{cc_i^*} = \delta_{cc_x^*}$, where $c_x^* = \arg\min_c d(\mathbf{x}, \mathbf{x}_c)$. Then $\sum_c p(c|\mathbf{x}) \log_2(p(c|\mathbf{x}) = 0$ and hence $I(\mathbf{x}, c) = H[p(c)]$.

## References

[1] W. Bialek and C. G. Callan and S. P. Strong, Phys. Rev. Lett. 77 (1996) 4693-4697, http://arxiv.org/abs/cond-mat/9607180

[2] W. Bialek in *Physics of bio-molecules and cells; École d'ete de physique théorique Les Houches Session LXXV* Eds.: H. Flyvbjerg, F. Jülicher, P. Ormos and F. David (2001) Springer-Verlag, pp.485–577, http://arxiv.org/abs/physics/0205030

[3] M. Blatt, S. Wiseman and E. Domany, Phys. Rev. Lett. 76 (1996) 3251-3254, http://arxiv.org/abs/cond-mat/9702072

[4] C. Fraley and A. Raftery, J. Am. Stat. Assoc. 97 (2002) 611-631.

[5] A. D. Gordon, *Classification*, (1999) Chapmann and Hall/CRC Press, London.

[6] P. Hall and E. J. Hannan, Biometrika 75, 4 (1988) 705-714.

[7] D. Horn and A. Gottlieb, Phys. Rev. Lett. 88 (2002) 018702, extended version: http://arxiv.org/abs/physics/0107063

[8] J. MacQueen in *Proc. 5th Berkeley Symp. Math. Statistics and Probability* Eds.: L.M.L Cam and J. Neyman (1967) University of California Press, pp. 281-297 (Vol. I)

[9] K. Rose, E. Gurewitz and G. C. Fox, Phys. Rev. Lett. 65 (1990) 945; and: K. Rose, Proceedings of the IEEE 86, 11 (1998) pp. 2210-2239.

[10] C. E. Shannon, Bell System Tech. J. 27, (1948). pp. 379-423, 623-656. See also: C. Shannon and W. Weaver, *The Mathematical Theory of Communication* (1963) University of Illinois Press

[11] P. Smyth, Statistics and Computing 10, 1 (2000) 63-72.

[12] S. Still and W. Bialek (2003, submitted), available at http://arxiv.org/abs/physics/0303011

[13] N. Tishby, F. Pereira and W. Bialek in *Proc. 37th Annual Allerton Conf.* Eds.: B. Hajek and R. S. Sreenivas (1999) University of Illinois, http://arxiv.org/abs/physics/0004057
